# Hierarchical Fisher Kernels for Longitudinal Data

**Zhengdong Lu**     **Todd K. Leen**
Dept. of Computer Science & Engineering
Oregon Health & Science University
Beaverton, OR 97006
luz@cs.utexas.edu,tleen@csee.ogi.edu

**Jeffrey Kaye**
Layton Aging & Alzheimer's Disease Center
Oregon Health & Science University
Portland, OR 97201
kaye@ohsu.edu

## Abstract

We develop new techniques for time series classification based on hierarchical Bayesian generative models (called *mixed-effect models*) and the Fisher kernel derived from them. A key advantage of the new formulation is that one can compute the Fisher information matrix despite varying sequence lengths and varying sampling intervals. This avoids the commonly-used ad hoc replacement of the Fisher information matrix with the identity which destroys the geometric invariance of the kernel. Our construction retains the geometric invariance, resulting in a kernel that is *properly invariant* under change of coordinates in the model parameter space. Experiments on detecting cognitive decline show that classifiers based on the proposed kernel out-perform those based on generative models and other feature extraction routines, and on Fisher kernels that use the identity in place of the Fisher information.

## 1   Introduction

Time series classification arises in diverse application. This paper develops new techniques based on hierarchical Bayesian generative models and the Fisher kernel derived from them. A key advantage of the new formulation is that, despite varying sequence lengths and sampling times, one can compute the Fisher information matrix. This avoids its common ad hoc replacement with the identity matrix. The latter strategy, common in the biological sequence literature [4], destroys the geometrical invariance of the kernel. Our construction retains the proper geometric structure, resulting in a kernel that is *properly invariant* under change of coordinates in the model parameter space.

This work was motivated by the need to classify clinical longitudinal data on human motor and psychometric test performance. Clinical studies show that at the population level progressive slowing of walking and the rate at which a subject can tap their fingers are predictive of cognitive decline years before its manifestation [1]. Similarly, performance on psychometric tests such as delayed recall of a story or word lists( tests *not* used in diagnosis), are predictive of cognitive decline [8]. An early predictor of cognitive decline for individual patients based on such longitudinal data would improve medical care and planning for assistance.

Our new Fisher kernels use mixed-effects models [6] as the generative process. These are hierarchical models that describe the population (consisting of many individuals) as a whole, and variations between individuals in the population. The population model parameters (called *fixed effects*), the covariance of the between-individual variability (the *random effects*), and the additive noise variance are fit by maximum likelihood. The overall population model together with the covariance of the random effects comprise a set of parameters for the prior on an individual subject model, so the fitting scheme is a hierarchical empirical Bayesian procedure.

**Data Description**   The data for this study was drawn from the Oregon Brain Aging Study (OBAS) [2], a longitudinal study spanning up to fifteen years with roughly yearly assessment of subjects. For our work, we grouped the subjects into two classes: those who remain cognitively healthy through the course of the study (denoted *normal*), and those who progress to mild cognitive impairment (MCI) or further to dementia (denoted *impaired*). Since we are interested in *prediction*, we retain only data taken *prior* to diagnosis of impairment. We use 97 subjects from the normal group and 46 from the group that becomes impaired. Motor task data included the time (denoted as seconds) and the number of steps (denoted as steps) to walk 9 meters, and the number of times the subject can tap their forefinger, both dominant (tappingD) and non-dominant hands (tappingN) in 10 seconds. Psychometric test data include delayed-recall, which measures the number of words from a list of 10 that the subject can recall one minute after hearing the list, and logical memory II in which the subject is graded on recall of a story told 15-20 minutes earlier.

## 2   Mixed-effect Models

### 2.1   Mixed-effect Regression Models

In this paper, we confine attention to parametric regression. Suppose there are $k$ individuals (indexed by $i = 1, \ldots, k$) contributing data to the sample, and we have observations $\{t_n^i, y_n^i\}$, $n = 1, \ldots, N^i$ as a function of time $t$ for individual $i$. The data are modeled as $y_n^i = f(t_n^i; \gamma^i) + \epsilon_n^i$, where $\gamma^i$ are the regression parameters and $\epsilon_n^i$ is zero-mean white Gaussian noise with (unknown) variance $\sigma^2$. The superscript on the model parameters $\gamma^i$ indicates that the regression parameters are different for each individual contributing to the population. Since the model parameters vary between individuals, it is natural to consider them generated by the sum of a fixed and a random piece: $\gamma^i = \alpha + \beta^i$, where $\beta^i$ (called the *random effect*), is assumed distributed $\mathcal{N}(0, \mathbf{D})$ with unknown covariance $\mathbf{D}$. The expected parameter vector $\alpha$, called the *fixed effect*, determines the model for the population as a whole, and the random effect $\beta^i$ accounts for the differences between individuals. This intuition is most precise for the case in which the model is linear in parameters

$$f(t; \gamma) = \gamma^T \Phi(t) = \alpha^T \Phi(t) + \beta^T \Phi(t) \tag{1}$$

where $\Phi(t) = [\phi_1(t), \phi_2(t), ..., \phi_d(t)]^T$ denotes a vector of basis functions[1]. We use $\mathcal{M} = \{\alpha, \mathbf{D}, \sigma\}$ to denote the mixed-effect model parameters. The feature values, observation times, and observation noise are

$$\mathbf{y}^i \equiv [y_1^i, \cdots, y_{N^i}^i]^T, \quad \mathbf{t}^i \equiv [t_1^i, \cdots, t_{N^i}^i]^T, \quad \epsilon^i \equiv [\epsilon_1^i, \cdots, \epsilon_{N^i}^i]^T \ .$$

### 2.2   Maximum Likelihood Fitting

Model fitting uses the entire collection of data $\{\mathbf{t}^i, \mathbf{y}^i\}$, $i = 1, \ldots, k$ to determine the parameters $\mathcal{M} = \{\alpha, \mathbf{D}, \sigma\}$ by maximum likelihood. The likelihood of the data $\{\mathbf{t}^i, \mathbf{y}^i\}$ given $\mathcal{M}$ is

$$p(\mathbf{y}^i; \mathbf{t}^i, \mathcal{M}) = \int p(\mathbf{y}^i | \beta^i; \mathbf{t}^i, \sigma) p(\beta^i | \mathcal{M}) d\beta^i \tag{2}$$

$$= (2\pi)^{-N^i/2} |\Sigma^i|^{-1/2} \exp((\mathbf{y}^i - \alpha^T \Phi(\mathbf{t}^i))^T (\Sigma^i)^{-1} (\mathbf{y}^i - \alpha^T \Phi^i(\mathbf{t}^i))) \tag{3}$$

where

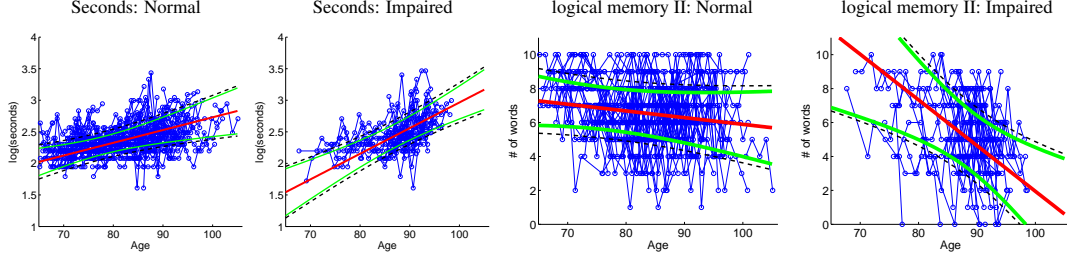

| Seconds: Normal | Seconds: Impaired | logical memory II: Normal | logical memory II: Impaired |

Figure 1: The fit mixed-effect models for two tests. In each panel, the red line stands for the fixed effect $\alpha^T \Phi(t)$. The two green lines stand for $\alpha^T \Phi(t) \pm \sqrt{\Phi^T(t)\mathbf{D}\Phi(t)}$, i.e., the population model $\pm$ the s.t.d. of the deviation from the uncertainty of the $\beta$. The black dash line is the s.t.d of the deviation when we consider the observation noise.

$$\Sigma^i = \sum_{n=1}^{N^i} \Phi(t_n^i)\mathbf{D}\Phi(t_n^i)^T + \sigma^2\mathbf{I}, \quad \text{and} \quad \Phi(\mathbf{t}^i) = [\Phi(t_1^i), \Phi(t_2^i), \cdots, \Phi(t_n^i)]^T.$$

The data likelihood for $\mathbf{Y} = \{\mathbf{y}^1, \mathbf{y}^2, \cdots, \mathbf{y}^k\}$ with $\mathbf{T} = \{\mathbf{t}^1, \mathbf{t}^2, \cdots, \mathbf{t}^k\}$ is then $p(\mathbf{Y}; \mathbf{T}, \mathcal{M}) = \prod_{i=1}^{k} p(\mathbf{y}^i | \mathbf{t}^i; \mathcal{M})$. The maximum likelihood values of $\{\alpha, \mathbf{D}, \sigma\}$ are found using the Expectation-Maximization algorithm [6] with $\{\beta^1, \beta^2, \cdots, \beta^k\}$ considered as the latent variable:

E-step: $\quad Q(\mathcal{M}, \mathcal{M}^g) = E_{\{\beta^i\}}(\log p(\mathbf{Y}, \{\beta^i\}; \mathbf{T}, \mathcal{M})|\mathbf{Y}; \mathbf{T}, \mathcal{M}^g)$ $\hfill$ (4)

M-step: $\quad \mathcal{M} = \arg\max_{\mathcal{M}} Q(\mathcal{M}, \mathcal{M}^g),$ $\hfill$ (5)

where $\mathcal{M}^g$ stands for the model parameters estimated in previous step, and the expectation in the E-step is with respect to the posterior distribution of on $\{\beta^i\}$ when $\mathbf{Y}$ is known and the model parameter is $\mathcal{M}^g$. For the linear mixed-effect model in Equation (1), the M-step can be given in a closed form. The details of the updating equations are given by Laird et al. [6].

We use the linear mixed-effect model with polynomial basis functions $\Phi(t) = [1, t]^T$. We trained separate mixed-effect models for each of the six measurements. For the four motor behavior measurements, we use the logarithm of data to reduce the skew of the residuals. Figure 1 shows the fit models for seconds and logical memory II, as the representatives of the six measurements. The plots show the fixed effect regression $\alpha^T\Phi(t)$ (red curve), and the standard deviations arising from the random effects (green curves) and measurement noise (dashed black curve, see caption). The data are the blue spaghetti plots. The plots confirm that subjects that become impaired deteriorate faster than those who remain healthy.

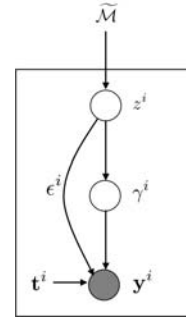

Figure 2: The graphical model of the mixture of mixed-effect models.

With multiple classes (or component subpopulations), it is natural to use a mixture of mixed-effect models. We have two components: one fit on the normal group (denoted $\mathcal{M}_0$) and one fit on impaired group (denoted $\mathcal{M}_1$), with $\mathcal{M}_m = \{\alpha_m, \mathbf{D}_m, \sigma_m\}, \quad m = 0, 1$. Here, we use $\widetilde{\mathcal{M}} = \{\pi_0, \mathcal{M}_0, \pi_1, \mathcal{M}_1\}$ to denote the parameters of this mixture, with $\pi_0$ and $\pi_1$ being the mixing proportions (prior) estimated from the training data. The overall generative process for any individual $(\mathbf{t}^i, \mathbf{y}^i)$ is summarized in Figure 2. Here $z^i \in \{0, 1\}$ is the latent variable indicating which model component is used to generate $\mathbf{y}^i$.

## 3 Hierarchical Fisher Kernel

### 3.1 Fisher Kernel Background

The Fisher kernel [4] provides a way to extract discriminative features from the generative model. For any $\theta$-parameterized model $p(x; \theta)$, the Fisher kernel between $x^i$ and $x^j$ is defined as

$$K(x^i, x^j) = (\nabla_\theta \log p(x^i; \theta))^T \mathbf{I}^{-1} \nabla_\theta \log p(x^j; \theta), \tag{6}$$

where $\mathbf{I}$ is the Fisher information matrix with the $(n, m)$ entry

$$\mathbf{I}_{n,m} = \int_x \frac{\partial \log p(x; \theta)}{\partial \theta_n} \frac{\partial \log p(x; \theta)}{\partial \theta_m} p(x; \theta) dx. \tag{7}$$

The kernel entry $K(x^i, x^j)$ can be viewed as the inner product of the natural gradient $\mathbf{I}^{-1} \nabla_\theta \log p(x; \theta)$ at $x^i$ and $x^j$ with metric $\mathbf{I}$, and is invariant to re-parametrization of $\theta$. Jaakkola and Haussler [4] prove that a linear classifier based on the Fisher kernel performs at least as well as the generative model.

### 3.2 Retaining the Fisher Information Matrix

In the bioinformatics literature [3] and for longitudinal data such as ours, $p(x^i; \theta)$ is different for each individual owing to different sequence lengths, and (for longitudinal data) different sampling times $\mathbf{t}^i$. The integral in Equation (7) must therefore include the distribution sequence lengths and observation times. Where only sequence lengths differ, an empirical average can be used. However where observation times are non-uniform and vary considerably between individuals (as is the case here), there is insufficient data to form an estimate by empirical averaging.

The usual response to the difficulty is to replace the Fisher information with the identity matrix [4]. This spoils the geometric structure, in particular the invariance of the the kernel $K(x^i, x^j)$ under change of coordinates in the model parameter space (model re-parameterization). This is a *significant flaw*: the coordinate system used to describe the model is immaterial and should not influence the value of $K(x^i, x^j)$. For probabilistic kernel regression, the choice of metric is immaterial in the limit of large training sets [4]. However for our application, which uses a support vector machine (SVM), we found the difference *cannot* be neglected.

In our case, replacing Fisher information matrix with the identity matrix is grossly unsuitable. For the mixed-effect model with polynomial basis functions the Fisher score components associated with higher order terms (such as slope and curvature) are far larger than the entries associated with lower order term (such as intercept). Without the proper normalization provided by the Fisher information matrix, the kernel will be dominated by higher order entries[2]. A principled extension of the Fisher kernel provided by our hierarchical model allows proper calculation of the Fisher information matrix.

### 3.3 Hierarchical Fisher Kernel

Our design of kernel is based on the generative hierarchy of mixture of mixed-effect models, in Figure 2. We notice that the individual-specific information $\mathbf{t}^i$ enter into this generative process at the last step, but the "latent" variables $\gamma^i$ and $z^i$ are drawn from the Gaussian mixture model (GMM) $\tilde{\Theta} = \{\pi_0, \alpha_0, \mathbf{D}_0, \pi_1, \alpha_1, \mathbf{D}_1\}$, with $p(z^i, \gamma^i; \tilde{\Theta}) = \pi_{z^i} p(\gamma_{z^i}; \alpha_{z^i}, \mathbf{D}_{z^i})$.

We can thus build a standard Fisher kernel for the latent variables, and use it to induce a kernel on the observed data. Denoting the latent variables by $v^i$, the Fisher kernel between $v^i$ and $v^j$ is

$$K(v^i, v^j) = (\nabla_\Theta \log p(v^i; \theta))^T (\mathbf{I}^v)^{-1} \nabla_\theta \log p(v^j; \Theta),$$

where the Fisher score $\nabla_{\tilde{\Theta}} \log p(v^i; \tilde{\Theta})$ is a column vector

$$\nabla_{\tilde{\Theta}} \log p(v^i; \tilde{\Theta}) = [\frac{\partial \log p}{\partial \pi_0}; \frac{\partial \log p}{\partial \alpha_0}; \frac{\partial \log p}{\partial \mathbf{D}_0}; \frac{\partial \log p}{\partial \pi_1}; \frac{\partial \log p}{\partial \alpha_1}; \frac{\partial \log p}{\partial \mathbf{D}_1}]^T,$$

and $\mathbf{I}^v$ is the well-defined Fisher information matrix for $v$:

$$\mathbf{I}^v_{n,m} = \int_v \frac{\partial \log p(v; \tilde{\Theta})}{\partial \tilde{\Theta}_n} \frac{\partial \log p(v; \tilde{\Theta})}{\partial \tilde{\Theta}_m} p(v|\tilde{\Theta}) dv. \tag{8}$$

The kernel for $\mathbf{y}^i$ and $\mathbf{y}^j$ is the expectation of $K(v^i, v^j)$ given the observation $\mathbf{y}^i$ and $\mathbf{y}^j$.

$$K(\mathbf{y}^i, \mathbf{y}^j) = E_{v^i, v^j}[K(v^i, v^j)| \mathbf{y}^i, \mathbf{y}^j; \mathbf{t}^i, \mathbf{t}^j, \widetilde{\mathcal{M}}] = \iint K(v^i, v^j) p(v^i|\mathbf{y}^i; \mathbf{t}^i, \widetilde{\mathcal{M}}) p(v^j|\mathbf{y}^j; \mathbf{t}^j, \widetilde{\mathcal{M}}) dv^i dv^j$$

With different choices of latent variable $v$, we have three kernel design strategies in the following subsections. This extension to the Fisher kernel, named hierarchical Fisher kernel (HFK), enables us to deal with time series with irregular sampling and different sequence lengths. To our knowledge it has not been reported elsewhere in the literature.

## Design A: $v^i = \gamma^i$

This kernel design marginalizes out the higher level variable $\{z^i\}$ and constructs Fisher kernel between the $\{\gamma^i\}$. This generative process is illustrated in Figure 3 (left panel), which is the same graphical model in Figure 2 with latent variable $z^i$ marginalized out[3]. The Fisher kernel for $\gamma$ is

$$K(\gamma^i, \gamma^j) = (\nabla_{\tilde{\Theta}} \log p(\gamma^i|\tilde{\Theta}))^T (\mathbf{I}^\gamma)^{-1} \nabla_{\tilde{\Theta}} \log p(\gamma^i|\tilde{\Theta}). \tag{9}$$

The kernel between $\mathbf{y}^i$ and $\mathbf{y}^j$ as the expectation of $K(\gamma^i, \gamma^j)$:

$$K(\mathbf{y}^i, \mathbf{y}^j) = E_{\gamma^i, \gamma^j}(K(\gamma^i, \gamma^j)| \mathbf{y}^i, \mathbf{y}^j; \mathbf{t}^i, \mathbf{t}^j, \widetilde{\mathcal{M}}) \tag{10}$$

$$= (\int \nabla_{\tilde{\Theta}} \log p(\gamma^i|\tilde{\Theta}) p(\gamma^i|\mathbf{y}^i; \mathbf{t}^i, \widetilde{\mathcal{M}}) d\gamma^i)^T (\mathbf{I}^\gamma)^{-1} \int \nabla_{\tilde{\Theta}} \log p(\gamma^j|\tilde{\Theta}) p(\gamma^j|\mathbf{y}^j; \mathbf{t}^j \widetilde{\mathcal{M}}) d\gamma^j. \tag{11}$$

The computational drawback is that the integral required to evaluate $\int \nabla_{\tilde{\Theta}} \log p(\gamma^j|\tilde{\Theta}) p(\gamma^j|\mathbf{y}^j; \mathbf{t}^j \widetilde{\mathcal{M}}) d\gamma^j$ and $\mathbf{I}^r$ do not have an analytical solution. In our experiments, we estimated the integral with Monte-Carlo sampling.

## Design B: $v^i = (z^i, \gamma^i)$

This design strategy takes both $\gamma^i$ and $z^i$ as joint latent variable and build a Fisher kernel for them. The generative process, as summarized in Figure 3 (middle panel), gives the probability for latent variables

$$p(z^i, \gamma^i; \tilde{\Theta}) = \pi_{z^i} p(\gamma_i; \alpha_{z^i}, \mathbf{D}_{z^i}).$$

The Fisher kernel for the joint variable $(\gamma^i, z^i)$ is

$$K((z^i, \gamma^i), (z^j, \gamma^j)) = (\nabla_{\tilde{\Theta}} \log p(z^i, \gamma^i; \tilde{\Theta}))^T (\mathbf{I}^{z,\gamma})^{-1} \nabla_{\tilde{\Theta}} \log p(z^i, \gamma^i; \tilde{\Theta}), \tag{12}$$

where $\mathbf{I}^{z,\gamma}$ is the Fisher information matrix associated with distribution $p(z, \gamma; \tilde{\Theta})$. It can be shown that

$$K((z^i, \gamma^i), (z^j, \gamma^j)) = \frac{1}{\pi_{z^i}} \delta(z^i, z^j)(1 + K_{z^i}(\gamma^i, \gamma^j))$$

where $K_m(\gamma^i, \gamma^j)$ is the Fisher kernel for $\gamma^i$ associated with component $m\,(= 0, 1)$

$$K_m(\gamma^i, \gamma^j) = (\nabla_{\Theta_m} \log p(\gamma^i; \alpha_m, \mathbf{D}_m))^T \mathbf{I}_m^{-1} \nabla_{\Theta_m} \log p(\gamma^i; \alpha_m, \mathbf{D}_m), \tag{13}$$

The kernel for $\mathbf{y}^i$ and $\mathbf{y}^j$ is defined similarly as in Design A:

$$K(\mathbf{y}^i, \mathbf{y}^j) \quad = \quad E_{z^i, \gamma^i, z^j, \gamma^j}(K((z^i, \gamma^i), (z^j, \gamma^j)) | \mathbf{y}^i, \mathbf{y}^j; \mathbf{t}^i, \mathbf{t}^j, \widetilde{\mathcal{M}}) \tag{14}$$

where the integral can be evaluated analytically.

**Design C:** $\widetilde{\mathcal{M}} = \mathcal{M}_m,\ m = 0, 1$

This design uses one mixed-effect component instead of the mixture as the generative model, as illustrated in Figure 3 (right panel). Although any single $\mathcal{M}_m$ is not a satisfying generative model for the whole population, the resulting kernel is still useful for classification as follows. For either model, $m = 0, 1$, the Fisher score for the $i^{th}$ individual $\nabla_{\Theta_m} \log p(\gamma^i; \Theta_m)$ describes how the probability $p(\gamma^i; \Theta_m)$ responds to the change of parameters $\Theta_m$. This is a discriminative feature vector since the likelihood of $\gamma_i$ for individuals from different group are likely to have different response to the change of parameters $\Theta_m$. The kernel between $\gamma^i$ and $\gamma^j$ is $K_m(\gamma^i, \gamma^j)$ defined in Equation (13). And then the kernel for $\mathbf{y}^i$ and $\mathbf{y}^j$:

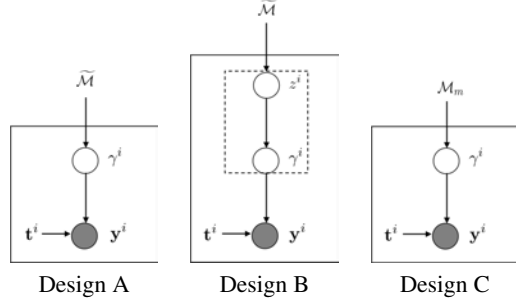

Figure 3: The graphical model of the mixture of mixed-effect models for Design A, B, and C.

$$K(\mathbf{y}^i, \mathbf{y}^j) \quad = \quad E_{\gamma^i, \gamma^j}(K(\gamma^i, \gamma^j) | \mathbf{y}^i, \mathbf{y}^j; \mathbf{t}^i, \mathbf{t}^j, \mathcal{M}_m) \tag{15}$$

Our experiments show that the kernel based on the impaired group is significantly better than others; we therefore use this kernel as the representative of Design C. It is easy to see that the designed kernel is a special case of Design A or Design B when $\pi_0 = 1$ and $\pi_1 = 0$.

### 3.4 Related Models

**Marginalized Kernel**    Our HFK is related to the marginalized kernel (MK) proposed by Tsuda et. al. [10]. MK uses a distribution with discrete latent variable $h$ (indicating the generating component) and observable $x$, which form a *complete data pair* $\mathbf{x} = (h, x)$. The kernel for observable $x^i$ and $x^j$ is defined as

$$\widetilde{K}(x^i, x^j) = \sum_{h^i} \sum_{h^j} P(h^i | x^i) P(h^j | x^j) \widetilde{K}(\mathbf{x}^i, \mathbf{x}^j) \tag{16}$$

where $\widetilde{K}(\mathbf{x}^i, \mathbf{x}^j)$ is the joint kernel for complete data. Tsuda et. al. [10] uses the form:

$$\widetilde{K}(\mathbf{x}^i, \mathbf{x}^j) = \delta(h^i, h^j) K_{h^i}(x^i, x^j), \tag{17}$$

where $K_{h^i}(x^i, x^j)$ is the pre-defined kernel for observables associated the $h^i$ generative component. Equation (17) says that $\widetilde{K}(\mathbf{x}^i, \mathbf{x}^j)$ takes the value of kernel defined for the $m^{th}$ component model if $x^i$ and $x^j$ are generated from the same component $h^i = h^j = m$; otherwise, $\widetilde{K}(\mathbf{x}^i, \mathbf{x}^j) = 0$. HFK can be viewed as a special case of the generalized marginalized kernel that allows continuous latent variables $h$. This is clear if we re-write Equation (16) as

$$\widetilde{K}(x^i, x^j) = E_{h^i, h^j}(\widetilde{K}(\mathbf{x}^i, \mathbf{x}^j) | x^i, x^j)$$

and view $\widetilde{K}(\mathbf{x}^i, \mathbf{x}^j)$ as a generalization of kernel between $h^i$ and $h^j$. Nevertheless HFK is different from the original work in [10], in that MK requires existing kernels for observable, such as $K_h(x^i, x^j)$ in Equation (17). In our problem setting, this kernel does not exist due to the different lengths of time series.

**Probability Product Kernel** We can get a family of kernels by employing various kernel designs of $K(v^i, v^j)$. The simplest example is to let $K(v^i, v^j) = \delta(z^i, z^j)$, which immediately leads to $K(\mathbf{y}^i, \mathbf{y}^j) = E_{v^i, v^j}(K(v^i, v^j)|\mathbf{y}^i, \mathbf{y}^j; \mathbf{t}^i, \mathbf{t}^j, \widetilde{\mathcal{M}}) = \sum_m P(z^i = m|\mathbf{y}^i; \mathbf{t}^i, \widetilde{\mathcal{M}}) P(z^j = m|\mathbf{y}^j; \mathbf{t}^j, \widetilde{\mathcal{M}})$, which is obviously related to the posterior probabilities of samples, and is essentially a special case of the probability product kernels [5] proposed by Jebara et. al.

## 4 Experiments

**Performance Evaluation** We use the empirical ROC curve (*detection rate* vs. *false alarm rate*) to evaluate classifiers. We compare different classifiers using the area under the curve (AUC), and calculate the statistical significance following the method given by Pepe [9]. We tested the classifiers on the five features: steps, seconds, tappingD, tappingN, and logical memory II. The results of delayed-recall are omitted, they are very close to those for logical memory II. The mixed-effect models for each feature were trained *separately* with order-1 polynomials (linear) as the basis functions. For each feature, the kernels are used in support vector machines (SVM) for classification, and the ROC is obtained by thresholding the classifier output with varying values. The classifiers are evaluated by leave-one-out cross-validation, the left-out sample consisting of an individual subject's complete time series (which is also held out of the fitting of the generative model).

**Classifiers** For comparison, we also examined the following two classifiers. First, we consider the *likelihood ratio test* based on mixed-effect models $\{\mathcal{M}_0, \mathcal{M}_1\}$. For any given observation $(\mathbf{t}, \mathbf{y})$, the likelihood that it is generated by mixed-effect model $\mathcal{M}_m$ is given by $p(\mathbf{y}; \mathbf{t}, \mathcal{M}_m)$, which is defined similarly as in Equation (3). The classification decision for a likelihood ratio classifier is made by thresholding the ratio $\frac{p(\mathbf{y}; \mathbf{t}, \mathcal{M}_0)}{p(\mathbf{y}; \mathbf{t}, \mathcal{M}_1)}$. Second, we consider a feature extraction routine independent of any generative model. We summarize each individual $i$ with the least-square fit coefficients for a $d$-degree polynomial regression model, denoted as $\mathbf{p}^i$. To get a reliable fitting we only consider the case $d = 1$ since many individuals only have four or five observations. We use the coefficients (normalized to their s.t.d.), denoted as $\hat{\mathbf{p}}^i$, as the feature vector, and build a RBF kernel $\mathbf{G}_{ij} = \exp(-\frac{||\hat{\mathbf{p}}^i - \hat{\mathbf{p}}^j||_2^2}{2s^2})$, where $s$ is the kernel width estimated with leave-one-out cross validation in our experiment. The obtained kernel matrix $\mathbf{G}$ will be referred to as *LSQ kernel*.

**Results** We first compare three HFK designs, using the ROC curves plotted in Figure 4 (upper row). On all four motor tests, Design A and Design B are very much comparable except on tappingD, on which Design A is marginally better than Design B with $p = 0.136$. Also on the motor tests, Design C is slightly but consistently better than other two designs. On logical memory II (story recall), the three designs have comparable performance. We thus use Design C as the representative of HFK, and compare it with the likelihood ratio classifier and SVM based on LSQ kernel, as shown in Figure 4 (lower row). On four motor test, the classifier based on HFK obviously out-performs the other two classifiers, and on logical memory II, the three classifiers have very much comparable performance.

## 5 Discussion

Fisher kernels derived from mixed-effect generative models retain the Fisher information matrix, and hence the proper invariance of the kernel under change of coordinates in the model parameter space. In additional experiments, classifiers constructed with the proper kernel out-perform those constructed with the identity matrix in place of the Fisher information on our data. For example, on seconds, the HKF (Deign C) achieves AUC = 0.7333, while the Fisher kernel computed with the identity matrix as metric on $p(\mathbf{y}^i; \mathbf{t}^i, \mathcal{M})$ achieves a AUC = 0.6873, with the $p$-value (Z-test) 0.0435.

Our classifiers built with Fisher kernels derived from mixed-effect models outperform those based solely on the generative model (using likelihood ratio tests) for the motor task data, and are comparable on the psychometric tests. The hierarchical kernels also produce better classifiers than a standard SVM using the coefficients of a least squares fit to the individual's data. This shows that the generative model provides real advantage for classification. The mixed-effect models capture both the population behavior (through $\alpha$), and the statistical variability of the individual subject models (through the covariance of $\beta$). Knowledge of

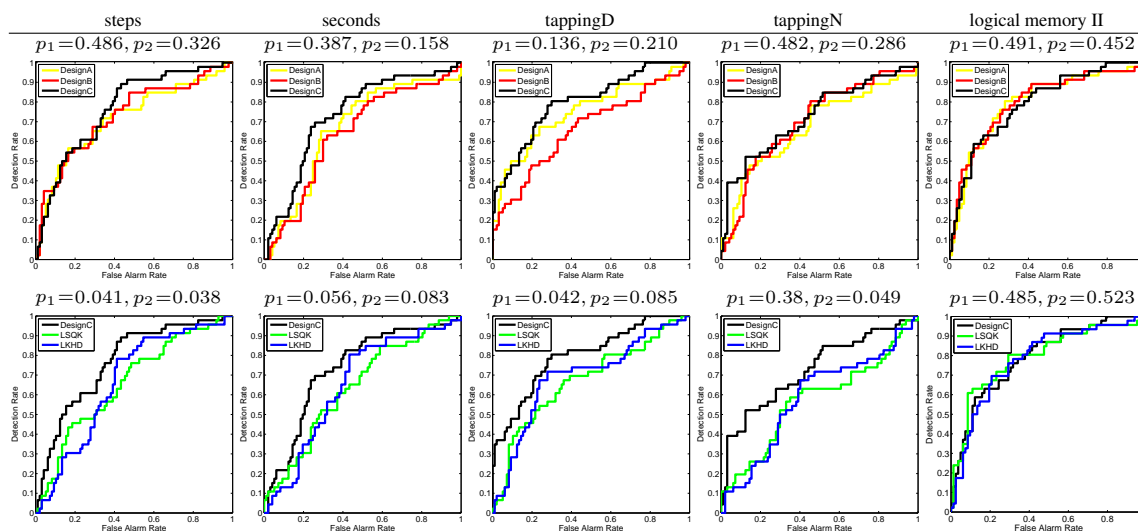

Figure 4: Comparison of classifiers. **Upper row:** Three HFK designs. The number in the parenthesis is the $p$-value (Z-test) for the null-hypothesis "the AUC of Classifier 1 is the same as the AUC of Classifier 2". **Upper row:** Three HKF designs. $p_1$: Design A vs. Design B, $p_2$: Design C vs. Design A; **Lower row:** HFK & other classifiers. $p_1$: Design C vs. Likelihood ratio, $p_2$: Design C vs. LSQ kernel.

the statistics of the subject variability is extremely important for classification: although not discussed here, classifiers based only on the population model ($\alpha$) perform far worse than those presented here [7].

### Acknowledgements

This work was supported by Intel Corp. under the OHSU BAIC award. Milar Moore and to Robin Guariglia of the Layton Aging & Alzheimer's Disease Center gave invaluable help with data from the Oregon Brain Aging Study. We thank Misha Pavel, Tamara Hayes, and Nichole Carlson for helpful discussion.

## Footnotes

[1]More generally, the fixed and random effects can be associated with different basis functions.

[2]Our experiments on the OBAS data show that replacing the Fisher information with the identity compromises classifier performance.

[3]Strictly speaking, we cannot sum out $z^i$ at this step since the group membership is used later in generating the observation noise. However this is a reasonable approximation since the noise variance from $\mathcal{M}_0$ and $\mathcal{M}_1$ are similar.

### References

[1] R. Camicioli, D. Howieson, B. Oken, G. Sexton, and J. Kaye. Motor slowing precedes cognitive impairment in the oldest old. *Neurology*, 50:1496–1498, 1998.

[2] M. Green, J. Kaye, and M. Ball. The Oregon brain aging study: Neuropathology accompanying healthy aging in the oldest old. *Neurology*, 54(1):105–113, 2000.

[3] T. Jaakkola, M. Diekhaus, and D. Haussler. Using the fisher kernel method to detect remote protein homologies. *7th Intell. Sys. Mol. Biol.*, pages 149–158, 1999.

[4] T. Jaakkola and D. Haussler. Exploiting generative models in discriminative classifiers. Technical report, Dept. of Computer Science, Univ. of California, 1998.

[5] T. Jebara, R. Kondor, and A. Howard. Probability product kernels. *Journal of Machine Learning Research*, 5:819–844, 2004.

[6] N. Laird and J. Ware. Random-effects models for longitudinal data. *Biometrics*, 38(4):963–974, 1982.

[7] Z. Lu. *Constrained Clustering and Cognitive Decline Detection*. PhD thesis, OHSU, 2008.

[8] S. Marquis, M. Moore, D. Howieson, G. Sexton, H. Payami, J. Kaye, and R. Camicioli. Independent predictors of cognitive decline in healthy elderly persons. *Arch. Neurol.*, 59:601–606, 2002.

[9] M. Pepe. *The Statistical Evaluation of Medical Tests for Classification and Prediction*. Oxford University Press, Oxford, 2003.

[10] K. Tsuda, T. Kin, and K. Asai. Marginalized kernels for biological sequences. *Bioinformatics*, 1(1):1–8, 2002.

